# Analysis of Empirical Bayesian Methods for Neuroelectromagnetic Source Localization

**David Wipf**[1], **Rey Ramírez**[2], **Jason Palmer**[1,2], **Scott Makeig**[2], **& Bhaskar Rao**[1] [*]
[1]Signal Processing and Intelligent Systems Lab
[2]Swartz Center for Computational Neuroscience
University of California, San Diego 92093
{dwipf,japalmer,brao}@ucsd.edu, {rey,scott}@sccn.ucsd.edu

## Abstract

The ill-posed nature of the MEG/EEG source localization problem requires the incorporation of prior assumptions when choosing an appropriate solution out of an infinite set of candidates. Bayesian methods are useful in this capacity because they allow these assumptions to be explicitly quantified. Recently, a number of empirical Bayesian approaches have been proposed that attempt a form of model selection by using the data to guide the search for an appropriate prior. While seemingly quite different in many respects, we apply a unifying framework based on automatic relevance determination (ARD) that elucidates various attributes of these methods and suggests directions for improvement. We also derive theoretical properties of this methodology related to convergence, local minima, and localization bias and explore connections with established algorithms.

## 1 Introduction

Magnetoencephalography (MEG) and electroencephalography (EEG) use an array of sensors to take EM field measurements from on or near the scalp surface with excellent temporal resolution. In both cases, the observed field is generated by the same synchronous, compact current sources located within the brain. Because the mapping from source activity configuration to sensor measurement is many to one, accurately determining the spatial locations of these unknown sources is extremely difficult. The relevant localization problem can be posed as follows: The measured EM signal is $B \in \Re^{d_b \times n}$, where $d_b$ equals the number of sensors and $n$ is the number of time points at which measurements are made. The unknown sources $S \in \Re^{d_s \times n}$ are the (discretized) current values at $d_s$ candidate locations distributed throughout the cortical surface. These candidate locations are obtained by segmenting a structural MR scan of a human subject and tesselating the gray matter surface with a set of vertices. $B$ and $S$ are related by the generative model

$$B = LS + \mathcal{E}, \tag{1}$$

where $L$ is the so-called lead-field matrix, the $i$-th column of which represents the signal vector that would be observed at the scalp given a unit current source at the $i$-th vertex with a fixed orientation (flexible orientations can be incorporated by including three columns per location, one for each directional component). Multiple methods based on the physical properties of the brain and Maxwell's equations are available for this computation. Finally, $\mathcal{E}$ is a noise term with columns drawn independently from $\mathcal{N}(0, \Sigma_\epsilon)$.

To obtain reasonable spatial resolution, the number of candidate source locations will necessarily be much larger than the number of sensors ($d_s \gg d_b$). The salient inverse problem then becomes the ill-posed estimation of these activity or source regions, which are reflected by the nonzero rows of the source estimate matrix $\hat{S}$. Because the inverse model is underdetermined, all efforts at source reconstruction are heavily dependent on prior assumptions, which in a Bayesian framework are embedded in the distribution $p(S)$. Such a prior is often considered to be fixed and known, as in the

---

[*]This work was supported by NSF grants DGE-0333451 and IIS-0613595.

case of minimum $\ell_2$-norm approaches, minimum current estimation (MCE) [6, 18], FOCUSS [2, 5], and sLORETA [10]. Alternatively, a number of empirical Bayesian approaches have been proposed that attempt a form of model selection by using the data to guide the search for an appropriate prior. Examples include variational Bayesian methods [14, 15], hierarchial covariance component models [4, 8, 11], and automatic relevance determination (ARD) [7, 9, 12, 13, 17]. While seemingly quite different in some respects, we present a generalized framework that encompasses many of these methods and points to connections between algorithms. We also analyze several theoretical properties of this framework related to computational/convergence issues, local minima, and localization bias. Overall, we envision that by providing a unifying perspective on these approaches, neuroelectromagnetic imaging practitioners will be better able to assess the relative strengths with respect to a particular application. This process also points to several promising directions for future research.

## 2 A Generalized Bayesian Framework for Source Localization

In this section, we present a general-purpose Bayesian framework for source localization. In doing so, we focus on the common ground between many of the methods discussed above. While derived using different assumptions and methodology, they can be related via the notion of automatic relevance determination [9] and evidence maximization [7].

To begin we involve the noise model from (1), which fully defines the assumed likelihood $p(B|S)$. While the unknown noise covariance can also be parameterized and estimated from the data, for simplicity we assume that $\Sigma_\epsilon$ is known and fixed. Next we adopt the following source prior for $S$:

$$p\left(S; \Sigma_s\right) = \mathcal{N}\left(0, \Sigma_s\right), \qquad \Sigma_s = \sum_{i=1}^{d_\gamma} \gamma_i C_i, \tag{2}$$

where the distribution is understood to apply independently to each column of $S$. Here $\boldsymbol{\gamma} = [\gamma_1, \ldots, \gamma_{d_\gamma}]^T$ is a vector of $d_\gamma$ nonnegative hyperparameters that control the relative contribution of each covariance basis matrix $C_i$, all of which we assume are fixed and known. The unknown hyperparameters can be estimated from the data by first integrating out the unknown sources $S$ giving

$$p(B; \Sigma_b) = \int p\left(B|S\right) p\left(S; \Sigma_s\right) dS = \mathcal{N}(0, \Sigma_b), \tag{3}$$

where $\Sigma_b = \Sigma_\epsilon + L\Sigma_s L^T$. A hyperprior $p(\boldsymbol{\gamma})$ can also be included if desired. This expression is then maximized with respect to the unknown hyperparameters, a process referred to as type-II maximum likelihood or evidence maximization [7, 9] or restricted maximum likelihood [4]. Thus the optimization problem shifts from finding the maximum a posteriori sources given a fixed prior to finding the optimal hyperparameters of a parameterized prior. Once these estimates are obtained (computational issues will be discussed in Section 2.1), a tractable posterior distribution $p(S|B; \hat{\Sigma}_s)$ exists in closed form, where $\hat{\Sigma}_s = \sum_i \hat{\gamma}_i C_i$. To the extent that the 'learned' prior $p(S; \hat{\Sigma}_s)$ is realistic, this posterior quantifies regions of significant current density and point estimates for the unknown sources can be obtained by evaluating the posterior mean

$$\hat{S} \triangleq \mathrm{E}\left[S|B; \hat{\Sigma}_s\right] = \hat{\Sigma}_s L^T \left(\Sigma_\epsilon + L\hat{\Sigma}_s L^T\right)^{-1} B. \tag{4}$$

The specific choice of the $C_i$'s is crucial and can be used to reflect any assumptions about the possible distribution of current sources. It is this selection, rather than the adoption of a covariance component model per se, that primarily differentiates the many different empirical Bayesian approaches and points to novel algorithms for future study. The optimization strategy adopted for computing $\hat{\boldsymbol{\gamma}}$, as well as the particular choice of hyperprior $p(\boldsymbol{\gamma})$, if any, can also be distinguishing factors.

In the simplest case, use of the single component $\Sigma_s = \gamma_1 C_1 = \gamma_1 I$ leads to a regularized minimum-$\ell_2$-norm solution. More interesting covariance component terms have been used to effect spatial smoothness, depth bias compensation, and candidate locations of likely activity [8, 11]. With regard to the latter, it has been suggested that prior information about a source location can be codified by including a $C_i$ term with all zeros except a patch of 1's along the diagonal signifying a location of probable source activity, perhaps based on fMRI data [11]. An associated hyperparameter $\gamma_i$ is then estimated to determine the appropriate contribution of this component to the overall prior covariance. The limitation of this approach is that we generally do not know, a priori, the regions

where activity is occurring with both high spatial and temporal resolution. Therefore, we cannot reliably known how to choose an appropriate location-prior term in many situations.

The empirical Bayesian solution to this dilemma, which amounts to a form of model selection, is to try out many different (or even all possible) combinations of location priors, and determine which one has the highest Bayesian evidence, i.e., maximizes $p(B; \Sigma_b)$ [7]. For example, if we assume the underlying currents are formed from a collection of dipolar point sources located at each vertex of the lead-field grid, then we may choose $\Sigma_s = \sum_{i=1}^{d_s} \gamma_i \boldsymbol{e}_i \boldsymbol{e}_i^T$, where each $\boldsymbol{e}_i$ is a standard indexing vector of zeros with a '1' for the $i$-th element (and so $C_i = \boldsymbol{e}_i \boldsymbol{e}_i^T$ encodes a prior preference for a single dipolar source at location $i$).[1] This specification for the prior involves the counterintuitive addition of an unknown hyperparameter for every candidate source location which, on casual analysis may seem prone to severe overfitting (in contrast to [11], which uses only one or two fixed location priors). However, the process of marginalization, or the integrating out of the unknown sources $S$, provides an extremely powerful regularizing effect, driving most of the unknown $\gamma_i$ to zero during the evidence maximization stage (more on this in Section 3). This ameliorates the overfitting problem and effectively reduces the space of possible active source locations by choosing a small relevant subset of location priors that optimizes the Bayesian evidence (hence ARD). With this 'learned' prior in place, a once ill-posed inverse problem is no longer untenable, with the posterior mean providing a good estimate of source activity. Such a procedure has been empirically successful in the context of neural networks [9], kernel machines [17], and multiple dipole fitting for MEG [12], a significant benefit to the latter being that the optimal number of dipoles need not be known a priori.

In contrast, to model sources with some spatial extent, we can choose $C_i = \boldsymbol{\psi}_i \boldsymbol{\psi}_i^T$, where each $\psi_i$ represents, for example, an $d_s \times 1$ geodesic neural basis vector that specifies an *a priori* weight location *and* activity extent [13]. In this scenario, the number of hyperparameters satisfies $d_\gamma = v d_s$, where $v$ is the number of scales we wish to examine in a multi-resolution decomposition, and can be quite large ($d_\gamma \approx 10^6$). As mentioned above, the ARD framework tests many priors corresponding to many hypotheses or beliefs regarding the locations and scales of the nonzero current activity within the brain, ultimately choosing the one with the highest evidence. The net result of this formulation is a source prior composed of a mixture of Gaussian kernels of varying scales. The number of mixture components, or the number of nonzero $\gamma_i$'s, is learned from the data and is naturally forced to be small (sparse). In general, the methodology is quite flexible and other prior specifications can be included as well, such as temporal and spectral constraints. But the essential ingredient of ARD, that marginalization and subsequent evidence maximization leads to a pruning of unsupported hypotheses, remains unchanged.

We turn now to empirical Bayesian procedures that incorporate variational methods. In [15], a plausible hierarchical prior is adopted that, unfortunately, leads to intractable integrations when computing the desired source posterior. This motivates the inclusion of a variational approximation that models the true posterior as a factored distribution over parameters at two levels of the prior hierarchy. While seemingly quite different, drawing on results from [1], we can show that the resulting cost function is exactly equivalent to standard ARD assuming $\Sigma_s$ is parameterized as

$$\Sigma_s = \sum_{i=1}^{d_s} \gamma_i \boldsymbol{e}_i \boldsymbol{e}_i + \sum_{j=1}^{d_s} \gamma_{(d_s+j)} \boldsymbol{\psi}_j \boldsymbol{\psi}_j^T, \tag{5}$$

and so $d_\gamma = 2d_s$. When fMRI data is available, it is incorporated into a particular inverse Gamma hyperprior on $\boldsymbol{\gamma}$, as is also commonly done with ARD methods [1]. Optimization is then performed using simple EM update rules.

In summary then, the general methods of [4, 8, 11] and [12, 13, 17] as well as the variational method of [15] are all identical with respect to their ARD-based cost functions; they differ only in which covariance components (and possibly hyperpriors) are used and in how optimization is performed as will be discussed below. In contrast, the variational model from [14] introduces an additional hierarchy to the ARD framework to explicitly model temporal correlations between sources which may be spatially separated.[2] Here it is assumed that $S$ can be decomposed with respect to $d_z$ *pre-*

*sources* via

$$S = WZ, \quad p(W; \Sigma_w) = \mathcal{N}(0, \Sigma_w), \ p(Z) = \mathcal{N}(0, I), \tag{6}$$

where $Z \in \Re^{d_z \times n}$ represents the pre-source matrix and $\Sigma_w$ is analogous to $\Sigma_s$. As stated in [14], direct application of ARD would involve integration over $W$ and $Z$ to find the hyperparameters $\gamma$ that maximize $p(B; \Sigma_b)$. While such a procedure is not analytically tractable, it remains insightful to explore the characteristics of this method were we able to perform the necessary computation. This allows us to relate the full model of [14] to standard ARD.

Interestingly, it can be shown that the first and second order statistics of the full prior (6) and the standard ARD prior (2) are equivalent (up to a constant factor), although higher-order moments will be different. However, as the number of pre-sources $d_z$ becomes large, multivariate central-limit-theorem arguments can be used to explicitly show that the distribution of $S$ converges to an identical Gaussian prior as ARD. So exact evaluation of the full model, which is espoused as the ideal objective were it feasible, approaches regular ARD when the number of pre-sources grows large. In practice, because the full model is intractable, a variational approximation is adopted similar to that proposed in [15]. In fact, if we assume the appropriate hyperprior on $\gamma$, then this correlated source method is essentially the same as the procedure from [15] but with an additional level in the approximate posterior factorization for handling the decomposition (6). This produces approximate posteriors on $W$ and $Z$ but the result cannot be integrated to form the posterior on $S$. However, the posterior mean of $W$, $\hat{W}$, is used as an estimate of the source correlation matrix (using $\hat{W}\hat{W}^T$) to substantially improve beamforming results that were errantly based on uncorrelated source models. Note however that this procedure implicitly uses the somewhat peculiar criteria of combining the posterior mean of $W$ with the prior on $Z$ to form an estimate of the distribution of $S$.

## 2.1 Computational Issues

The primary objective of ARD is to maximize the evidence $p(B; \Sigma_b)$ with respect to $\gamma$ or equivalently, to minimize

$$\mathcal{L}(\gamma) \triangleq -\log p(B; \Sigma_b) \equiv n \log |\Sigma_b| + \text{trace} \left[ B^T \Sigma_b^{-1} B \right]. \tag{7}$$

In [4], a restricted maximum likelihood (ReML) approach is proposed for this optimization, which utilizes what amounts to EM-based updates. This method typically requires a nonlinear search for each M-step and does not guarantee that the estimated covariance is positive definite. While shown to be successful in estimating a handful of hyperparameters in [8, 11], this could potentially be problematic when very large numbers of hyperparameters are present. For example, in several toy problems (with $d_\gamma$ large) we have found that a fraction of the hyperparameters obtained can be negative-valued, inconsistent with our initial premise.

As such, we present three alternative optimization procedures that extend the methods from [7, 12, 15, 17] to the arbitrary covariance model discussed above and guarantee that $\gamma_i \geq 0$ for all $i$. Because of the flexibility this allows in constructing $\Sigma_s$, and therefore $\Sigma_b$, some additional notation is required to proceed. A new decomposition of $\Sigma_b$ is defined as

$$\Sigma_b = \Sigma_\epsilon + L \left( \sum_{i=1}^{d_\gamma} \gamma_i C_i \right) L^T = \Sigma_\epsilon + \sum_{i=1}^{d_\gamma} \gamma_i \widetilde{L}_i \widetilde{L}_i^T, \tag{8}$$

where $\widetilde{L}_i \widetilde{L}_i^T \triangleq L C_i L^T$ with $r_i \triangleq \text{rank}(\widetilde{L}_i \widetilde{L}_i^T) \leq d_b$. Also, using commutative properties of the trace operator, $\mathcal{L}(\gamma)$ only depends on the data $B$ through the $d_b \times d_b$ sample correlation matrix $BB^T$. Therefore, to reduce the computational burden, we replace $B$ with a matrix $\widetilde{B} \in \Re^{d_b \times \text{rank}(B)}$ such that $\widetilde{B}\widetilde{B}^T = BB^T$. This removes any per-iteration dependency on $n$, which can potentially be large, without altering that actual cost function.

By treating the unknown sources as hidden data, an update can be derived for the $(k+1)$-th iteration

$$\gamma_i^{(k+1)} = \frac{1}{nr_i} \left\| \gamma_i^{(k)} \widetilde{L}_i^T \left( \Sigma_b^{(k)} \right)^{-1} \widetilde{B} \right\|_{\mathcal{F}}^2 + \frac{1}{r_i} \text{trace} \left[ \gamma_i^{(k)} I - \gamma_i^{(k)} \widetilde{L}_i^T \left( \Sigma_b^{(k)} \right)^{-1} \widetilde{L}_i \gamma_i^{(k)} \right], \tag{9}$$

which reduces to the algorithm from [15] given the appropriate simplifying assumptions on the form of $\Sigma_s$ and some additional algebraic manipulations. It is also equivalent to ReML with a

different effective computation for the M-step. By casting the update rules in this way and noting that off-diagonal elements of the second term need not be computed, the per-iteration cost is at most $\mathrm{O}\left(d_b^2 \sum_{i=1}^{d_\gamma} r_i\right) \leq \mathrm{O}\left(d_b^3 d_\gamma\right)$. This expense can be significantly reduced still further in cases where different pseudo lead-field components, e.g., some $\widetilde{L}_i$ and $\widetilde{L}_j$, contain one or more columns in common. This situation occurs if we desire to use the geodesic basis functions with flexible orientation constraints, as opposed to the fixed orientations assumed above. In general, the linear dependence on $d_\gamma$ is one of the attractive aspects of this method, effectively allowing for extremely large numbers of hyperparameters and covariance components.

The problem then with (9) is not the per-iteration complexity but the convergence rate, which we have observed to be prohibitively slow in practical situations with high-resolution lead-field matrices and large numbers of hyperparameters. The only reported localization results using this type of EM algorithm are from [15], where a relatively low resolution lead-field matrix is used in conjunction with a simplifying heuristic that constrains some of the hyperparameter values. However, to avoid these types of constraints, which can potentially degrade the quality of source estimates, a faster update rule is needed. To this end, we modified the procedure of [7], which involves taking the gradient of $\mathcal{L}(\gamma)$ with respect to $\gamma$, rearranging terms, and forming the fixed-point update

$$\gamma_i^{(k+1)} = \frac{\gamma_i^{(k)}}{n} \left\| \widetilde{L}_i^T \left(\Sigma_b^{(k)}\right)^{-1} \widetilde{B} \right\|_{\mathcal{F}}^2 \left(\operatorname{trace}\left[\widetilde{L}_i^T \left(\Sigma_b^{(k)}\right)^{-1} \widetilde{L}_i\right]\right)^{-1}. \tag{10}$$

The complexity of each iteration is the same as before, only now the convergence rate can be orders of magnitude faster. For example, given $d_b = 275$ sensors, $n = 1000$ observation vectors, and using a pseudo lead-field with 120,000 unique columns and an equal number of hyperparameters, requires approximately 5-10 mins. runtime using Matlab code on a PC to completely converge. The EM update does not converge after 24 hours. Example localization results using (10) demonstrate the ability to recover very complex source configurations with variable spatial extent [13].

Unlike the EM method, one criticism of (10) is that there currently exists no proof that it represents a descent function, although we have never observed it to increase (7) in practice. While we can show that (10) is equivalent to iteratively solving a particular min-max problem in search of a saddle point, provable convergence is still suspect. However, a similar update rule can be derived that is both significantly faster than EM *and* is proven to produce $\gamma$ vectors such that $\mathcal{L}\left(\gamma^{(k+1)}\right) \leq \mathcal{L}\left(\gamma^{(k)}\right)$ for every iteration $k$. Using a dual-form representation of $\mathcal{L}(\gamma)$ that leads to a more tractable auxiliary cost function, this update is given by

$$\gamma_i^{(k+1)} = \frac{\gamma_i^{(k)}}{\sqrt{n}} \left\| \widetilde{L}_i^T \left(\Sigma_b^{(k)}\right)^{-1} \widetilde{B} \right\|_{\mathcal{F}} \left(\operatorname{trace}\left[\widetilde{L}_i^T \left(\Sigma_b^{(k)}\right)^{-1} \widetilde{L}_i\right]\right)^{-1/2}. \tag{11}$$

Details of the derivation can be found in [20].

Finally, the correlated source method from [14] can be incorporated into the general ARD framework as well using update rules related to the above; however, because all off-diagonal terms are required by this method, the iterations now scale as $\left(\sum_i r_i\right)^2$ in the general case. This quadratic dependence can be prohibitive in applications with large numbers of covariance components.

## 2.2 Relationship with Other Bayesian Methods

As a point of comparison, we now describe how ARD can be related to alternative Bayesian-inspired approaches such as the sLORETA paradigm [10] and the iterative FOCUSS source localization algorithm [5]. The connection is most transparent when we substitute the prior covariance $\Sigma_s = \sum_{i=1}^{d_s} \gamma_i e_i e_i^T = \operatorname{diag}[\gamma]$ into (10), giving the modified update

$$\gamma_i^{(k+1)} = \left\| \gamma_i^{(k)} \ell_i^T \left(\Sigma_\epsilon + L\Gamma^{(k)}L^T\right)^{-1} B \right\|_2^2 \left(nR_{ii}^{(k)}\right)^{-1}, \quad R^{(k)} \triangleq \Gamma^{(k)}L^T \left(\Sigma_\epsilon + L\Gamma^{(k)}L^T\right)^{-1} L, \tag{12}$$

where $\Gamma \triangleq \operatorname{diag}[\gamma]$, $\ell_i$ is the $i$-th column of $L$, and $R^{(k)}$ is the effective resolution matrix given the hyperparameters at the current iteration. The $j$-th column of $R$ (called a point-spread function) equals the source estimate obtained using (4) when the true source is a unit dipole at location $j$ [16].

Continuing, if we assume that initialization of ARD occurs with $\gamma^{(0)} = \mathbf{1}$ (as is customary), then the hyperparameters produced after a *single* iteration of ARD are equivalent to computing the sLORETA

estimate for standardized current density power [10] (this assumes fixed orientation constraints). In this context, the inclusion of $R$ as a normalization factor helps to compensate for depth bias, which is the propensity for deep current sources within the brain to be underrepresented at the scalp surface [10, 12]. So ARD can be interpreted as a recursive refinement of what amounts to the non-adaptive, linear sLORETA estimate.

As a further avenue for comparison, if we assume that $R = I$ for all iterations, then the update (12) is nearly the same as the FOCUSS iterations modified to simultaneously handle multiple observation vectors [2]. The only difference is the factor of $n$ in the denominator in the case of ARD, but this can be offset by an appropriate rescaling of the FOCUSS $\lambda$ trade-off parameter (analogous to $\Sigma_\epsilon$). Therefore, ARD can be viewed in some sense as taking the recursive FOCUSS update rules and including the sLORETA normalization that, among other things, allows for depth bias compensation.

Thus far, we have focused on similarities in update rules between the ARD formulation (restricted to the case where $\Sigma_s = \Gamma$) and sLORETA and FOCUSS. We now switch gears and examine how the general ARD cost function relates to that of FOCUSS and MCE and suggests a useful generalization of both approaches. Recall that the evidence maximization procedure upon which ARD is based involves integrating out the unknown *sources* before optimizing the hyperparameters $\boldsymbol{\gamma}$. However, if some $p(\boldsymbol{\gamma})$ is assumed for $\boldsymbol{\gamma}$, then we could just as easily do the opposite: namely, we can integrate out the *hyperparameters* and then maximize $S$ directly, thus solving the MAP estimation problem

$$\max_S \int p\left(B|S\right) p\left(S; \Sigma_s\right) p(\boldsymbol{\gamma}) d\boldsymbol{\gamma} \quad \equiv \quad \min_{\{S:S=\sum_i A_i \widetilde{S}_i\}} \|B - LS\|_{\Sigma_\epsilon^{-1}}^2 + \sum_{i=1}^{d_\gamma} g\left(\|\widetilde{S}_i\|_{\mathcal{F}}\right), \quad (13)$$

where each $A_i$ is derived from the $i$-th covariance component such that $C_i = A_i A_i^T$, and $g(\cdot)$ is a function dependent on $p(\boldsymbol{\gamma})$. For example, when $p(\boldsymbol{\gamma})$ is a noninformative Jeffreys prior, then $g(x) = \log x$ and (13) becomes a generalized form of the FOCUSS cost function (and reduces to the exact FOCUSS cost when $A_i = \boldsymbol{e}_i$ for all $i$). Likewise, when an exponential prior chosen, then $g(x) = x$ and we obtain a generalized version of MCE. In both cases, multiple simultaneous constraints (e.g., flexible dipole orientations, spatial smoothing, etc.) can be naturally handled and, if desired, the noise covariance $\Sigma_\epsilon$ can be seamlessly estimated as well (see [3] for a special case of the latter in the context of kernel regression). This addresses many of the concerns raised in [8] pertaining to existing MAP methods. Additionally, as with ARD, source components that are not sufficiently important in representing the observed data are pruned; however, the undesirable discontinuities in standard FOCUSS or MCE source estimates across time, which previously have required smoothing using heuristic measures [6], do not occur when using (13). This is because sparsity is only encouraged *between* components due to the concavity of $g(\cdot)$, but not *within* components where the Frobenius norm operator promotes smooth solutions [2]. All of these issues, as well as efficient ARD-like update rules for optimizing (13), are discussed in [20].

## 3    General Properties of ARD Methods

ARD methods maintain several attributes that make them desirable candidates for source localization. For example, unlike most MAP procedures, the ARD cost function is often invariant to lead-field column normalizations, which only affect the implicit initialization that is used or potentially the selection of the $C_i$'s. In contrast, MCE produces a different globally minimizing solution for every normalization scheme. As such, ARD is considerably more robust to the particular heuristic used for this task and can readily handle deep current sources.

Previously, we have claimed that the ARD process naturally forces excessive/irrelevant hyperparameters to converge to zero, thereby reducing model complexity. While this observation has been verified empirically by ourselves and others in various application settings, there has been relatively little corroborating theoretical evidence, largely because of the difficulty in analyzing the potentially multimodal, non-convex ARD cost function. As such, we provide the following result:

**Result 1.** Every local minimum of the generalized ARD cost function (7) is achieved at a solution with at most $\mathrm{rank}(B)d_b \leq d_b^2$ nonzero hyperparameters.

The proof follows from a result in [19] and the fact that the ARD cost only depends on the $\mathrm{rank}(B)$ matrix $BB^T$. Result 1 comprises a worst-case bound that is only tight in very nuanced situations; in practice, for any reasonable value of $\Sigma_\epsilon$, the number of nonzero hyperparameters is typically much smaller than $d_b$. The bound holds for all $\Sigma_\epsilon$, including $\Sigma_\epsilon = 0$, indicating that some measure of

hyperparameter pruning, and therefore covariance component pruning, is built into the ARD framework irrespective of the noise-based regularization. Moreover, the number of nonzero hyperparameters decreases monotonically to zero as $\Sigma_\epsilon$ is increased. And so there is always some $\Sigma_\epsilon = \Sigma'_\epsilon$ sufficiently large such that all hyperparameters converge to exactly zero. Therefore, we can be reasonable confident that the pruning mechanism of ARD is not merely an empirical phenomena. Nor is it dependent on a particular sparse hyperprior, since the ARD cost from (7) implicitly assumes a flat (uniform) hyperprior.

The number of observation vectors $n$ also plays an important role in shaping ARD solutions. Increasing $n$ has two primary benefits: (i) it facilitates convergence to the global minimum (as opposed to getting stuck in a suboptimal extrema) and (ii), it improves the quality of this minimum by mitigating the effects of noise [20]. With perfectly correlated (spatially separated) sources, primarily only the later benefit is in effect. For example, with low noise and perfectly correlated sources, the estimation problem reduces to an equivalent problem with $n = 1$, so the local minima profile of the cost function does not improve with increasing $n$. Of course standard ARD can still be very effective in this scenario [13]. In contrast, geometric arguments can be made to show that uncorrelated sources with large $n$ offer the best opportunity for local minima avoidance. However, when strong correlations are present as well as high noise levels, the method of [14] (which explicitly attempts to model correlations) could offer a worthwhile alternative, albeit at a high computational cost.

Further theoretical support for ARD is possible in the context of localization bias assuming simple source configurations. For example, substantial import has been devoted to quantifying localization bias when estimating a single dipolar source. Recently it has been shown, both empirically [10] and theoretically [16], that sLORETA has zero location bias under this condition at high SNR. Viewed then as an iterative enhancement of sLORETA as described in Section 2.2, the question naturally arises whether ARD methods retain this desirable property. In fact, it can be shown that this is indeed the case in two general situations. We assume that the lead-field matrix $L$ represents a sufficiently high sampling of the source space such that any active dipole aligns with some lead-field column. Unbiasedness can also be shown in the continuous case for both sLORETA and ARD, but the discrete scenario is more straightforward and of course more relevant to any practical task.

**Result 2.** Assume that $\Sigma_s$ includes (among others) $d_s$ covariance components of the form $C_i = e_i e_i^T$. Then in the absence of noise (high SNR), ARD has provably zero localization bias when estimating a single dipolar source, regardless of the value of $n$.

If we are willing to tolerate some additional assumptions, then this result can be significantly expanded. For example, multiple dipolar sources can be localized with zero bias if they are perfectly uncorrelated (orthogonal) across time and assuming some mild technical conditions [20]. This result also formalizes the notion, mentioned above, that ARD performs best with uncorrelated sources. Turning to the more realistic scenario where noise is present gives the following:

**Result 3.** Let $\Sigma_s$ be constructed as above and assume the noise covariance matrix $\Sigma_\epsilon$ is known up to a scale factor. Then given a single dipolar source, in the limit as $n$ becomes large the ARD cost function is unimodal, and a source estimate with zero localization bias achieves the global minimum.

For most reasonable lead-fields and covariance components, this global minimum will be unique, and so the unbiased solution will be found as in the noiseless case. As for proofs, all the theoretical results pertaining to localization bias in this section follow from local minima properties of ML covariance component estimates. While details have been deferred to [20], the basic idea is that if the outerproduct $BB^T$ can be expressed as some non-negative linear combination of the available covariance components, then the ARD cost function is unimodal and $\Sigma_b = n^{-1}BB^T$ at any minimizing solution. This $\Sigma_b$ in turn produces unbiased source estimates in a variety of situations.

While theoretical results of this kind are admittedly limited, other iterative Bayesian schemes in fact fail to exhibit similar performance. For example, all of the MAP-based focal algorithms we are aware of, including FOCUSS and MCE methods, provably maintain a localization bias in the general setting, although in particular cases they may not exhibit one. (Also, because of the additional complexity involved, it is still unclear whether the correlated source method of [14] satisfies a similar result.) When we move to more complex source configurations with possible correlations and noise, theoretical results are not available; however, empirical tests provide a useful means of comparison. For example, given a $275 \times 40,000$ lead-field matrix constructed from an MR scan and assuming fixed orientation constraints and a spherical head model, ARD using $\Sigma_s = \text{diag}[\boldsymbol{\gamma}]$ and $n = 1$

(equivalent to having perfectly correlated sources) consistently maintains zero empirical localization bias when estimating up to 15-20 dipoles, while sLORETA starts to show a bias with only a few.

## 4 Discussion

The efficacy of modern empirical Bayesian techniques and variational approximations make them attractive candidates for source localization. However, it is not always transparent how these methods relate nor which should be expected to perform best in various situations. By developing a general framework around the notion of ARD, deriving several theoretical properties, and showing connections between algorithms, we hope to bring an insightful perspective to these techniques.

## Footnotes

[1] Here we assume dipoles with orientations constrained to be orthogonal to the cortical surface; however, the method is easily extended to handle unconstrained dipoles.

[2] Although standard ARD does not explicitly model correlated sources that are spatially separated, it still works well in this situation (see Section 3) and can reflect such correlations via the inferred posterior mean.

## References

[1] C. M. Bishop and M. E. Tipping, "Variational relevance vector machines," *Proc. 16th Conf. Uncertainty in Artificial Intelligence*, 2000.

[2] S.F. Cotter, B.D. Rao, K. Engan, and K. Kreutz-Delgado, "Sparse solutions to linear inverse problems with multiple measurement vectors," *IEEE Trans. Sig. Proc.*, vol. 53, no. 7, 2005.

[3] M.A.T. Figueiredo, "Adaptive sparseness using Jeffreys prior," *Advances in Neural Information Processing Systems 14*, MIT Press, 2002.

[4] K. Friston, W. Penny, C. Phillips, S. Kiebel, G. Hinton, and J. Ashburner, "Classical and Bayesian inference in neuroimaging: Theory," *NeuroImage*, vol. 16, 2002.

[5] I.F. Gorodnitsky, J.S. George, and B.D. Rao, "Neuromagnetic source imaging with FOCUSS: A recursive weighted minimum norm algorithm," *J. Electroencephalography and Clinical Neurophysiology*, vol. 95, no. 4, 1995.

[6] M. Huang, A. Dale, T. Song, E. Halgren, D. Harrington, I. Podgorny, J. Canive, S. Lewis, and R. Lee, "Vector-based spatial-temporal minimum $\ell_1$-norm solution for MEG," *NeuroImage*, vol. 31, 2006.

[7] D.J.C. MacKay, "Bayesian interpolation," *Neural Computation*, vol. 4, no. 3, 1992.

[8] J. Mattout, C. Phillips, W.D. Penny, M.D. Rugg, and K.J. Friston, "MEG source localization under multiple constraints: An extended Bayesian framework," *NeuroImage*, vol. 30, 2006.

[9] R.M. Neal, *Bayesian Learning for Neural Networks*, Springer-Verlag, New York, 1996.

[10] R.D. Pascual-Marqui, "Standardized low resolution brain electromagnetic tomography (sLORETA): Technical details," *Methods and Findings in Experimental and Clinical Pharmacology*, vol. 24, no. Suppl D, 2002.

[11] C. Phillips, J. Mattout, M.D. Rugg, P. Maquet, and K.J. Friston, "An empirical Bayesian solution to the source reconstruction problem in EEG," *NeuroImage*, vol. 24, 2005.

[12] R.R. Ramírez, *Neuromagnetic Source Imaging of Spontaneous and Evoked Human Brain Dynamics*, PhD thesis, New York University, 2005.

[13] R.R. Ramírez and S. Makeig, "Neuroelectromagnetic source imaging using multiscale geodesic neural bases and sparse Bayesian learning," *12th Conf. Human Brain Mapping*, 2006.

[14] M. Sahani and S.S. Nagarajan, "Reconstructing MEG sources with unknown correlations," *Advances in Neural Information Processing Systems 16*, MIT Press, 2004.

[15] M. Sato, T. Yoshioka, S. Kajihara, K. Toyama, N. Goda, K. Doya, and M. Kawato, "Hierarchical Bayesian estimation for MEG inverse problem," *NeuroImage*, vol. 23, 2004.

[16] K. Sekihara, M. Sahani, and S.S. Nagarajan, "Localization bias and spatial resolution of adaptive and non-adaptive spatial filters for MEG source reconstruction," *NeuroImage*, vol. 25, 2005.

[17] M.E. Tipping, "Sparse Bayesian learning and the relevance vector machine," *J. Machine Learning Research*, vol. 1, 2001.

[18] K. Uutela, M. Hämäläinen, and E. Somersalo, "Visualization of magnetoencephalographic data using minimum current estimates," *NeuroImage*, vol. 10, 1999.

[19] D.P. Wipf and B.D. Rao, "Sparse Bayesian learning for basis selection," *IEEE Trans. Sig. Proc.*, vol. 52, no. 8, 2004.

[20] D.P. Wipf and R.R. Ramírez and J.A. Palmer and S. Makeig and B.D. Rao, *Automatic Relevance Determination for Source Localization with MEG and EEG Data*, Technical Report, University of California, San Diego, 2006.
